# Optimistic Optimization of a Deterministic Function without the Knowledge of its Smoothness

**Rémi Munos**
SequeL project, INRIA Lille – Nord Europe, France
`remi.munos@inria.fr`

## Abstract

We consider a global optimization problem of a deterministic function $f$ in a semi-metric space, given a finite budget of $n$ evaluations. The function $f$ is assumed to be locally smooth (around one of its global maxima) with respect to a semi-metric $\ell$. We describe two algorithms based on optimistic exploration that use a hierarchical partitioning of the space at all scales. A first contribution is an algorithm, DOO, that requires the knowledge of $\ell$. We report a finite-sample performance bound in terms of a measure of the quantity of near-optimal states. We then define a second algorithm, SOO, which does not require the knowledge of the semi-metric $\ell$ under which $f$ is smooth, and whose performance is almost as good as DOO optimally-fitted.

## 1 Introduction

We consider the problem of finding a good approximation of the maximum of a function $f : \mathcal{X} \to \mathbb{R}$ using a finite budget of evaluations of the function. More precisely, we want to design a sequential exploration strategy of the search space $\mathcal{X}$, i.e. a sequence $x_1, x_2, \ldots, x_n$ of states of $\mathcal{X}$, where each $x_t$ may depend on previously observed values $f(x_1), \ldots, f(x_{t-1})$, such that at round $n$ (computational budget), the algorithms $\mathcal{A}$ returns a state $x(n)$ with highest possible value. The performance of the algorithm is evaluated by the loss

$$r_n = \sup_{x \in \mathcal{X}} f(x) - f(x(n)). \tag{1}$$

Here the performance criterion is the accuracy of the recommendation made after $n$ evaluations to the function (which may be thought of as calls to a black-box model). This criterion is different from usual bandit settings where the cumulative regret $(n \sup_{x \in \mathcal{X}} f(x) - \sum_{t=1}^{n} f(x(t)))$ measures how well the algorithm succeeds in selecting states with good values while exploring the search space. The loss criterion (1) is closer to the simple regret defined in the bandit setting [BMS09, ABM10].

Since the literature on global optimization is huge, we only mention the works that are closely related to our contribution. The approach followed here can be seen as an optimistic sampling strategy where, at each round, we explore the space where the function could be the largest, given the knowledge of previous evaluations. A large body of algorithmic work has been developed using branch-and-bound techniques [Neu90, Han92, Kea96, HT96, Pin96, Flo99, SS00], such as Lipschitz optimization where the function is assumed to be globally Lipschitz. Our first contribution with respect to (w.r.t.) this literature is to considerably weaken the Lipschitz assumption usually made and consider only a locally one-sided Lipschitz assumption around the maximum of $f$. In addition, we do not require the space to be a metric space but only to be equipped with a semi-metric.

The optimistic strategy has been recently intensively studied in the bandit literature, such as in the UCB algorithm [ACBF02] and the many extensions to tree search [KS06, CM07] (with application

to computer-go [GWMT06]), planning [HM08, BM10, BMSB11], and Gaussian process optimiza-
tion [SKKS10]. The case of Lipschitz (or relaxed) assumption in a metric spaces is considered in
[Kle04, AOS07] and more recently in [KSU08, BMSS08, BMSS11], and in the case of unknown
Lipschitz constant, see [BSY11, Sli11] (where they assume a bound on the Hessian or another re-
lated parameter).

Compared to this literature, our contribution is the design and analysis of two algorithms: (1) A first
algorithm, Deterministic Optimistic Optimization (DOO), that requires the knowledge of the semi-
metric $\ell$ for which $f$ is locally smooth around its maximum. A loss bound is provided (in terms of
the near-optimality dimension of $f$ under $\ell$) in a more general setting that previously considered.
(2) A second algorithm, Simultaneous Optimistic Optimization (SOO), that does not require the
knowledge of $\ell$. We show that SOO performs almost as well as DOO optimally-fitted.

## 2   Assumptions about the hierarchical partition and the function

Our optimization algorithms will be implemented by resorting to a hierarchical partitioning of the
space $\mathcal{X}$, which is given to the algorithms. More precisely, we consider a set of partitions of $\mathcal{X}$ at
all scales $h \geq 0$: for any integer $h$, $\mathcal{X}$ is partitioned into a set of $K^h$ sets $X_{h,i}$ (called cells), where
$0 \leq i \leq K^h - 1$. This partitioning may be represented by a $K$-ary tree structure where each cell
$X_{h,i}$ corresponds to a node $(h, i)$ of the tree (indexed by its depth $h$ and index $i$), and such that each
node $(h, i)$ possesses $K$ children nodes $\{(h + 1, i_k)\}_{1 \leq k \leq K}$. In addition, the cells of the children
$\{X_{h+1,i_k}, 1 \leq k \leq K\}$ form a partition of the parent's cell $X_{h,i}$. The root of the tree corresponds
to the whole domain $\mathcal{X}$ (cell $X_{0,0}$). To each cell $X_{h,i}$ is assigned a specific state $x_{h,i} \in X_{h,i}$ where
$f$ may be evaluated.

We now state 4 assumptions: Assumptions 1 is about the semi-metric $\ell$, Assumption 2 is about the
smoothness of the function w.r.t. $\ell$, and Assumptions 3 and 4 are about the shape of the hierarchical
partition w.r.t. $\ell$.

**Assumption 1** (Semi-metric). *We assume that $\ell : \mathcal{X} \times \mathcal{X} \to \mathbb{R}^+$ is such that for all $x, y \in \mathcal{X}$, we
have $\ell(x, y) = \ell(y, x)$ and $\ell(x, y) = 0$ if and only if $x = y$.*

Note that we do not require that $\ell$ satisfies the triangle inequality (in which case, $\ell$ would be a
metric). An example of a metric space is the Euclidean space $\mathbb{R}^d$ with the metric $\ell(x, y) = \|x - y\|$
(Euclidean norm). Now consider $\mathbb{R}^d$ with $\ell(x, y) = \|x - y\|^\alpha$, for some $\alpha > 0$. When $\alpha \leq 1$, then
$\ell$ is also a metric, but whenever $\alpha > 1$ then $\ell$ does not satisfy the triangle inequality anymore, and is
thus a semi-metric only.

**Assumption 2** (Local smoothness of $f$). *There exists at least a global optimizer $x^* \in \mathcal{X}$ of $f$ (i.e.,
$f(x^*) = \sup_{x \in \mathcal{X}} f(x)$) and for all $x \in \mathcal{X}$,*

$$f(x^*) - f(x) \leq \ell(x, x^*). \tag{2}$$

This condition guarantees that $f$ does not decrease too fast around (at least) one global optimum $x^*$
(this is a sort of a locally one-sided Lipschitz assumption).

Now we state the assumptions about the hierarchical partitions.

**Assumption 3** (Bounded diameters). *There exists a decreasing sequence $\delta(h) > 0$, such that for
any depth $h \geq 0$, for any cell $X_{h,i}$ of depth $h$, we have $\sup_{x \in X_{h,i}} \ell(x_{h,i}, x) \leq \delta(h)$.*

**Assumption 4** (Well-shaped cells). *There exists $\nu > 0$ such that for any depth $h \geq 0$, any cell $X_{h,i}$
contains a $\ell$-ball of radius $\nu\delta(h)$ centered in $x_{h,i}$.*

## 3   When the semi-metric $\ell$ is known

In this Section, we consider the setting where Assumptions 1-4 hold for a specific semi-metric $\ell$,
and that **the semi-metric $\ell$ is known from the algorithm**.

### 3.1   The DOO Algorithm

The Deterministic Optimistic Optimization (DOO) algorithm described in Figure 1 uses explicitly
the knowledge of $\ell$ (through the use of $\delta(h)$). DOO builds incrementally a tree $\mathcal{T}_t$ for $t = 1 \dots n$, by

```
Initialization: $\mathcal{T}_1 = \{(0,0)\}$ (root node)
for $t = 1$ to $n$ do
    Select the leaf $(h, j) \in \mathcal{L}_t$ with maximum $b_{h,j} \overset{\text{def}}{=} f(x_{h,j}) + \delta(h)$ value.
    Expand this node: add to $\mathcal{T}_t$ the $K$ children of $(h, j)$
end for
Return $x(n) = \arg\max_{(h,i)\in\mathcal{T}_n} f(x_{h,i})$
```

Figure 1: Deterministic optimistic optimization (DOO) algorithm.

selecting at each round $t$ a leaf of the current tree $\mathcal{T}_t$ to expand. Expanding a leaf means adding its $K$ children to the current tree (this corresponds to splitting the cell $X_{h,j}$ into $K$ sub-cells). We start with the root node $\mathcal{T}_1 = \{(0,0)\}$. We write $\mathcal{L}_t$ the leaves of $\mathcal{T}_t$ (set of nodes whose children are not in $\mathcal{T}_t$), which are the set of nodes that can be expanded at round $t$.

This algorithm is called optimistic because it expands at each round a cell that may contain the optimum of $f$, based on the information about (i) the previously observed evaluations of $f$, and (ii) the knowledge of the local smoothness property (2) of $f$ (since $\ell$ is known). The algorithm computes the b-values $b_{h,j} \overset{\text{def}}{=} f(x_{h,j}) + \delta(h)$ of all nodes $(h, j)$ of the current tree $\mathcal{T}_t$ and select the leaf with highest b-value to expand next. It returns the state $x(n)$ with highest evaluation.

## 3.2 Analysis of DOO

Note that Assumption 2 implies that the b-value of any cell containing $x^*$ upper bounds $f^*$, i.e., for any cell $X_{h,i}$ such that $x^* \in X_{h,i}$,

$$b_{h,i} = f(x_{h,i}) + \delta(h) \geq f(x_{h,i}) + \ell(x_{h,i}, x^*) \geq f^*.$$

As a consequence, a node $(h, i)$ such that $f(x_{h,i}) + \delta(h) < f^*$ will never be expanded (since at any time $t$, the b-value of such a node will be dominated by the b-value of the leaf containing $x^*$). We deduce that DOO only expands nodes of the set $I \overset{\text{def}}{=} \cup_{h\geq 0} I_h$, where

$$I_h \overset{\text{def}}{=} \{\text{nodes } (h, i) \text{ such that } f(x_{h,i}) + \delta(h) \geq f^*\}.$$

In order to derive a loss bound we now define a measure of the quantity of near-optimal states, called *near-optimality dimension*. This measure is closely related to similar measures introduced in [KSU08, BMSS08]. For any $\varepsilon > 0$, let us write $\mathcal{X}_\varepsilon \overset{\text{def}}{=} \{x \in \mathcal{X}, f(x) \geq f^* - \varepsilon\}$ the set of $\varepsilon$-optimal states.

**Definition 1** (*Near-optimality dimension*). *The near-optimality dimension is the smallest $d \geq 0$ such that there exists $C > 0$ such that for any $\varepsilon > 0$, the maximal number of disjoint $\ell$-balls of radius $\nu\varepsilon$ and center in $\mathcal{X}_\varepsilon$ is less than $C\varepsilon^{-d}$.*

Note that $d$ is not an intrinsic property of $f$: it characterizes both $f$ and $\ell$ (since we use $\ell$-balls in the packing of near-optimal states), and also depend on $\nu$. We now bound the number of nodes in $I_h$.

**Lemma 1.** *We have $|I_h| \leq C\delta(h)^{-d}$.*

*Proof.* From Assumption 4, each cell $(h, i)$ contains a ball of radius $\nu\delta(h)$ centered in $x_{h,i}$, thus if $|I_h| = |\{x_{h,i} \in \mathcal{X}_{\delta(h)}\}|$ exceeded $C\delta(h)^{-d}$, this would mean that there exists more than $C\delta(h)^{-d}$ disjoint $\ell$-balls of radius $\nu\delta(h)$ with center in $\mathcal{X}_{\delta(h)}$, which contradicts the definition of $d$. $\qquad\square$

We now provide our loss bound for DOO.

**Theorem 1.** *Let us write $h(n)$ the smallest integer $h$ such that $C\sum_{l=0}^{h} \delta(l)^{-d} \geq n$. Then the loss of DOO is bounded as $r_n \leq \delta(h(n))$.*

*Proof.* Let $(h_{\max}, j)$ be the deepest node that has been expanded by the algorithm up to round $n$. We known that DOO only expands nodes in the set $I$. Now, among all node expansion strategies of the set of expandable nodes $I$, the uniform strategy is the one which minimizes the depth of the resulting tree. From the definition of $h(n)$ and from Lemma 1, we have

$$\sum_{l=0}^{h(n)-1} |I_l| \leq C \sum_{l=0}^{h(n)-1} \delta(l)^{-d} < n,$$

thus the maximum depth of the uniform strategy is at least $h(n)$, and we deduce that $h_{\max} \geq h(n)$. Now since node $(h_{\max}, j)$ has been expanded, we have that $(h_{\max}, j) \in I$, thus

$$f(x(n)) \geq f(x_{h_{\max}, j}) \geq f^* - \delta(h_{\max}) \geq f^* - \delta(h(n)). \qquad \square$$

**Remark 1.** *This bound is in terms of the number of expanded nodes $n$. The actual number of function evaluations is $Kn$ (since each expansion generates $K$ children that need to be evaluated).*

Now, let us make the bound more explicit when the diameter $\delta(h)$ of the cells decreases exponentially fast with their depth (this case is rather general as illustrated in the examples described next, as well as in the discussion in [BMSS11]).

**Corollary 1.** *Assume that $\delta(h) = c\gamma^h$ for some constants $c > 0$ and $\gamma < 1$. If the near-optimality of $f$ is $d > 0$, then the loss decreases polynomially fast: $r_n \leq c^{\frac{d+1}{d}} \left(1 - \gamma^d\right)^{-1/d} C^{1/d} n^{-1/d}$. Now, if $d = 0$, then the loss decreases exponentially fast: $r_n \leq c\gamma^{(n/C)-1}$.*

*Proof.* From Theorem 1, whenever $d > 0$ we have $n \leq C \sum_{l=0}^{h(n)} \delta(l)^{-d} = cC \frac{\gamma^{-d(h(n)+1)}-1}{\gamma^{-d}-1}$, thus $\gamma^{-dh(n)} \geq \frac{n}{cC}\left(1 - \gamma^d\right)$, from which we deduce that $r_n \leq \delta(h(n)) \leq c\gamma^{h(n)} \leq c^{\frac{d+1}{d}}\left(1 - \gamma^d\right)^{-1/d} C^{1/d} n^{-1/d}$. Now, if $d = 0$ then $n \leq C \sum_{l=0}^{h(n)} \delta(l)^{-d} = C(h(n) + 1)$, and we deduce that the loss is bounded as $r_n \leq \delta(h(n)) = c\gamma^{(n/C)-1}$. $\qquad \square$

### 3.3 Examples

**Example 1:** Let $\mathcal{X} = [-1, 1]^D$ and $f$ be the function $f(x) = 1 - \|x\|_\infty^\alpha$, for some $\alpha \geq 1$. Consider a $K = 2^D$-ary tree of partitions with (hyper)-squares. Expanding a node means splitting the corresponding square in $2^D$ squares of half length. Let $x_{h,i}$ be the center of $X_{h,i}$.

Consider the following choice of the semi metric: $\ell(x, y) = \|x - y\|_\infty^\beta$, with $\beta \leq \alpha$. We have $\delta(h) = 2^{-h\beta}$ (recall that $\delta(h)$ is defined in terms of $\ell$), and $\nu = 1$. The optimum of $f$ is $x^* = 0$ and $f$ satisfies the local smoothness property (2). Now let us compute its near-optimality dimension. For any $\varepsilon > 0$, $\mathcal{X}_\varepsilon$ is the $L_\infty$-ball of radius $\varepsilon^{1/\alpha}$ centered in 0, which can be packed by $\left(\frac{\varepsilon^{1/\alpha}}{\varepsilon^{1/\beta}}\right)^D$ $L_\infty$-balls of diameter $\varepsilon$ (since a $L_\infty$-balls of diameter $\varepsilon$ is a $\ell$-ball of diameter $\varepsilon^{1/\beta}$). Thus the near-optimality dimension is $d = D(1/\beta - 1/\alpha)$ (and the constant $C = 1$). From Corollary 1 we deduce that (i) when $\alpha > \beta$, then $d > 0$ and in this case, $r_n = O\left(n^{-\frac{1}{D}\frac{\alpha\beta}{\alpha-\beta}}\right)$. And (ii) when $\alpha = \beta$, then $d = 0$ and the loss decreases exponentially fast: $r_n \leq 2^{1-n}$.

It is interesting to compare this result to a uniform sampling strategy (i.e., the function is evaluated at the set of points on a uniform grid), which would provide a loss of order $n^{-\alpha/D}$. We observe that DOO is better than uniform whenever $\alpha < 2\beta$ and worse when $\alpha > 2\beta$.

This result provides some indication on how to choose the semi-metric $\ell$ (thus $\beta$), which is a key ingredient of the DOO algorithm (since $\delta(h) = 2^{-h\beta}$ appears in the b-values): $\beta$ should be as close as possible to the true (but unknown) $\alpha$ (which can be seen as a local smoothness order of $f$ around its maximum), but never larger than $\alpha$ (otherwise $f$ does not satisfy the local smoothness property (2)).

**Example 2:** The previous analysis generalizes to any function which is locally equivalent to $\|x - x^*\|^\alpha$, for some $\alpha > 0$ (where $\|\cdot\|$ is any norm, e.g., Euclidean, $L_\infty$, or $L_1$), around a global maximum $x^*$ (among a set of global optima assumed to be finite). That is, we assume that there exists constants $c_1 > 0$, $c_2 > 0$, $\eta > 0$, such that

$$\begin{aligned}
f(x^*) - f(x) &\leq c_1 \|x - x^*\|^\alpha, && \text{for all } x \in \mathcal{X}, \\
f(x^*) - f(x) &\geq c_2 \|x - x^*\|^\alpha, && \text{for all } \|x - x^*\| \leq \eta.
\end{aligned}$$

Let $\mathcal{X} = [0,1]^D$. Again, consider a $K = 2^D$-ary tree of partitions with (hyper)-squares. Let $\ell(x,y) = c\|x - y\|^\beta$ with $c_1 \le c$ and $\beta \le \alpha$ (so that $f$ satisfies (2)). For simplicity we do not make explicit all the constants using the $O$ notation for convenience (the actual constants depend on the choice of the norm $\|\cdot\|$). We have $\delta(h) = O(2^{-h\beta})$. Now, let us compute the near-optimality dimension. For any $\varepsilon > 0$, $\mathcal{X}_\varepsilon$ is included in a ball of radius $(\varepsilon/c_2)^{1/\alpha}$ centered in $x^*$, which can be packed by $O\big(\frac{\varepsilon^{1/\alpha}}{\varepsilon^{1/\beta}}\big)^D$ $\ell$-balls of diameter $\varepsilon$. Thus the near-optimality dimension is $d = D(1/\beta - 1/\alpha)$, and the results of the previous example apply (up to constants), i.e. for $\alpha > \beta$, then $d > 0$ and $r_n = O\big(n^{-\frac{1}{D}\frac{\alpha\beta}{\alpha-\beta}}\big)$. And when $\alpha = \beta$, then $d = 0$ and one obtains the exponential rate $r_n = O(2^{-\alpha(n/C-1)})$.

We deduce that the behavior of the algorithm depends on our knowledge of the local smoothness (i.e. $\alpha$ and $c_1$) of the function around its maximum. Indeed, if this smoothness information is available, then one should defined the semi-metric $\ell$ (which impacts the algorithm through the definition of $\delta(h)$) to match this smoothness (i.e. set $\beta = \alpha$) and derive an exponential loss rate. Now if this information is unknown, then one should underestimate the true smoothness (i.e. by choosing $\beta \le \alpha$) and suffer a loss $r_n = O\big(n^{-\frac{1}{D}\frac{\alpha\beta}{\alpha-\beta}}\big)$, rather than overestimating it ($\beta > \alpha$) since in this case, (2) may not hold anymore and there is a risk that the algorithm converges to a local optimum (thus suffering a constant loss).

### 3.4 Comparison with previous works

**Optimistic planning:**   The deterministic planning problem described in [HM08] considers an optimistic approach for selecting the first action of a sequence $x$ that maximizes the sum of discounted rewards. We can easily cast their problem in our setting by considering the space $\mathcal{X}$ of the set of infinite sequences of actions. The metric $\ell(x,y)$ is $\gamma^{h(x,y)}/(1-\gamma)$, where $h(x,y)$ is the length of the common initial actions between the sequences $x$ and $y$, and $\gamma$ is the discount factor. It is easy to show that the function $f(x)$, defined as the discounted sum of rewards along the sequence $x$ of actions, is Lipschitz w.r.t. $\ell$ and thus satisfies (2). Their algorithm is very close to DOO: it expands a node of the tree (finite sequence of actions) with highest upper-bound on the possible value. Their regret analysis makes use of a quantity of near-optimal sequences, from which they define $\kappa \in [1, K]$ that can be seen as the branching factor of the set of nodes $I$ that can be expanded. This measure is related to our near-optimality dimension by $\kappa = \gamma^{-d}$. Corollary 1 implies directly that the loss bound is $r_n = O(n^{-\frac{\log 1/\gamma}{\log \kappa}})$ which is the result reported in [HM08].

**HOO and Zooming algorithms:**   The DOO algorithm can be seen as a deterministic version of the HOO algorithm of [BMSS11] and is also closely related to the Zooming algorithm of [KSU08]. Those works consider the case of noisy evaluations of the function ($\mathcal{X}$-armed bandit setting), which is assumed to be weakly Lipschitz (slightly stronger than our Assumption 2). The bounds reported in those works are (for the case of exponentially decreasing diameters considered in their work and in our Corollary 1) on the cumulative regret $R_n = O(n^{\frac{d+1}{d+2}})$, which translates into the loss considered here as $r_n = O(n^{-\frac{1}{d+2}})$, where $d$ is the near-optimality dimension (or the closely defined zooming dimension). We conclude that a deterministic evaluation of the function enables to obtain a much better polynomial rate $O(n^{-1/d})$ when $d > 0$, and even an exponential rate when $d = 0$ (Corollary 1).

In the next section, we address the problem of an unknown semi-metric $\ell$, which is the main contribution of the paper.

## 4   When the semi-metric $\ell$ is unknown

We now consider the setting where Assumptions 1-4 hold for some semi-metric $\ell$, but **the semi-metric $\ell$ is unknown**. The hierarchical partitioning of the space is still given, but since $\ell$ is unknown, one cannot use the diameter $\delta(h)$ of the cells to design upper-bounds, like in DOO.

The question we wish to address is: If $\ell$ is unknown, is it possible to implement an optimistic algorithm with performance guarantees? We provide a positive answer to this question and in addition we show that we can be **almost as good as an algorithm that would know $\ell$, for the best possible $\ell$ satisfying Assumptions 1-4.**

> The maximum depth function $t \mapsto h_{\max}(t)$ is a parameter of the algorithm.
> **Initialization:** $\mathcal{T}_1 = \{(0,0)\}$ (root node). Set $t = 1$.
> **while** True **do**
>     Set $v_{\max} = -\infty$.
>     **for** $h = 0$ to $\min(\mathrm{depth}(\mathcal{T}_t), h_{\max}(t))$ **do**
>         Among all leaves $(h,j) \in \mathcal{L}_t$ of depth $h$, select $(h,i) \in \arg\max_{(h,j)\in\mathcal{L}_t} f(x_{h,j})$
>         **if** $f(x_{h,i}) \geq v_{\max}$ **then**
>             Expand this node: add to $\mathcal{T}_t$ the $K$ children $(h+1, i_k)_{1 \leq k \leq K}$
>             Set $v_{\max} = f(x_{h,i})$, Set $t = t+1$
>             **if** $t = n$ **then Return** $x(n) = \arg\max_{(h,i)\in\mathcal{T}_n} x_{h,i}$
>         **end if**
>     **end for**
> **end while**.

Figure 2: Simultaneous Optimistic Optimization (SOO) algorithm.

## 4.1 The SOO algorithm

The idea is to expand at each round simultaneously all the leaves $(h,j)$ for which there exists a semi-metric $\ell$ such that the corresponding upper-bound $f(x_{h,j}) + \sup_{x\in X_{h,j}} \ell(x_{h,j}, x)$ would be the highest. This is implemented by expanding at each round at most a leaf per depth, and a leaf is expanded only if it has the largest value among all leaves of same or lower depths. The Simultaneous Optimistic Optimization (SOO) algorithm is described in Figure 2.

The SOO algorithm takes as parameter a function $t \rightarrow h_{\max}(t)$ which forces the tree to a maximal depth of $h_{\max}(t)$ after $t$ node expansions. Again, $\mathcal{L}_t$ refers to the set of leaves of $\mathcal{T}_t$.

## 4.2 Analysis of SOO

All previously relevant quantities such as the diameters $\delta(h)$, the sets $I_h$, and the near-optimality dimension $d$ depend on the unknown semi-metric $\ell$ (which is such that Assumptions 1-4 are satisfied).

At time $t$, let us write $h_t^*$ the depth of the deepest expanded node in the branch containing $x^*$ (an optimal branch). Let $(h_t^*+1, i^*)$ be an optimal node of depth $h_t^*+1$ (i.e., such that $x^* \in X_{h_t^*+1, i^*}$). Since this node has not been expanded yet, any node $(h_t^*+1, i)$ of depth $h_t^*+1$ that is later expanded, before $(h_t^*+1, i^*)$ is expanded, is $\delta(h_t^*+1)$-optimal. Indeed, $f(x_{h_t^*+1, i}) \geq f(x_{h_t^*+1, i^*}) \geq f^* - \delta(h_t^*+1)$. We deduce that once an optimal node of depth $h$ is expanded, it takes at most $|I_{h+1}|$ node expansions at depth $h+1$ before the optimal node of depth $h+1$ is expanded. From that simple observation, we deduce the following lemma.

**Lemma 2.** *For any depth* $0 \leq h \leq h_{\max}(t)$*, whenever* $t \geq (|I_0| + |I_1| + \cdots + |I_h|)h_{\max}(t)$*, we have* $h_t^* \geq h$.

*Proof.* We prove it by induction. For $h = 0$, we have $h_t^* \geq 0$ trivially. Assume that the proposition is true for all $0 \leq h \leq h_0$ with $h_0 < h_{\max}(t)$. Let us prove that it is also true for $h_0 + 1$. Let $t \geq (|I_0| + |I_1| + \cdots + |I_{h_0+1}|)h_{\max}(t)$. Since $t \geq (|I_0| + |I_1| + \cdots + |I_{h_0}|)h_{\max}(t)$ we know that $h_t^* \geq h_0$. So, either $h_t^* \geq h_0 + 1$ in which case the proof is finished, or $h_t^* = h_0$. In this latter case, consider the nodes of depth $h_0 + 1$ that are expanded. We have seen that as long as the optimal node of depth $h_0 + 1$ is not expanded, any node of depth $h_0 + 1$ that is expanded must be $\delta(h_0+1)$-optimal, i.e., belongs to $I_{h_0+1}$. Since there are $|I_{h_0+1}|$ of them, after $|I_{h_0+1}|h_{\max}(t)$ node expansions, the optimal one must be expanded, thus $h_t^* \geq h_0 + 1$. □

**Theorem 2.** *Let us write* $h(n)$ *the smallest integer* $h$ *such that*

$$C h_{\max}(n) \sum_{l=0}^{h} \delta(l)^{-d} \geq n. \tag{3}$$

*Then the loss is bounded as*

$$r_n \leq \delta\big(\min(h(n), h_{\max}(n) + 1)\big). \tag{4}$$

*Proof.* From Lemma 1 and the definition of $h(n)$ we have

$$h_{\max}(n) \sum_{l=0}^{h(n)-1} |I_l| \leq C h_{\max}(n) \sum_{l=0}^{h(n)-1} \delta(l)^{-d} < n,$$

thus from Lemma 2, when $h(n) - 1 \leq h_{\max}(n)$ we have $h_n^* \geq h(n) - 1$. Now in the case $h(n) - 1 > h_{\max}(n)$, since the SOO algorithm does not expand nodes beyond depth $h_{\max}(n)$, we have $h_n^* = h_{\max}(n)$. Thus in all cases, $h_n^* \geq \min(h(n) - 1, h_{\max}(n))$.

Let $(h, j)$ be the deepest node in $\mathcal{T}_n$ that has been expanded by the algorithm up to round $n$. Thus $h \geq h_n^*$. Now, from the definition of the algorithm, we only expand a node when its value is larger than the value of all the leaves of equal or lower depths. Thus, since the node $(h, j)$ has been expanded, its value is at least as high as that of the optimal node $(h_n^* + 1, i^*)$ of depth $h_n^* + 1$ (which has not been expanded, by definition of $h_n^*$). Thus

$$f(x(n)) \geq f(x_{h,j}) \geq f(x_{h_n^*+1,i^*}) \geq f^* - \delta(h_n^* + 1) \geq f^* - \delta(\min(h(n), h_{\max}(n) + 1)). \quad \square$$

**Remark 2.** *This result appears very surprising: although the semi-metric $\ell$ is not known, the performance is almost as good as for DOO (see Theorem 1) which uses the knowledge of $\ell$. The main difference is that the maximal depth $h_{\max}(n)$ appears both as a multiplicative factor in the definition of $h(n)$ in (3) and as a threshold in the loss bound (4). Those two appearances of $h_{\max}(n)$ defines a tradeoff between deep (large $h_{\max}$) versus broad (small $h_{\max}$) types of exploration. We now illustrate the case of exponentially decreasing diameters.*

**Corollary 2.** *Assume that $\delta(h) = c\gamma^h$ for some $c > 0$ and $\gamma < 1$. Consider the two cases:*

- *The near-optimality $d > 0$. Let the depth function $h_{\max}(t) = t^\varepsilon$, for some $\varepsilon > 0$ arbitrarily small. Then, for $n$ large enough (as a function of $\varepsilon$) the loss of SOO is bounded as:*

$$r_n \leq c^{\frac{d+1}{d}} \left( \frac{C}{1 - \gamma^d} \right)^{1/d} n^{-\frac{1-\varepsilon}{d}}.$$

- *The near-optimality $d = 0$. Let the depth function $h_{\max}(t) = \sqrt{t}$. Then the loss of SOO is bounded as:*

$$r_n \leq c\gamma^{\sqrt{n}\min(1/C,1)-1}.$$

*Proof.* From Theorem 1, when $d > 0$ we have

$$n \leq C h_{\max}(n) \sum_{l=0}^{h(n)} \delta(l)^{-d} = cC h_{\max}(n) \frac{\gamma^{-d(h(n)+1)} - 1}{\gamma^{-d} - 1}$$

thus for the choice $h_{\max}(n) = n^\varepsilon$, we deduce $\gamma^{-dh(n)} \geq \frac{n^{1-\varepsilon}}{cC}(1 - \gamma^d)$. Thus $h(n)$ is logarithmic in $n$ and for $n$ large enough (as a function of $\varepsilon$), $h(n) \leq h_{\max}(n) + 1$, thus

$$r_n \leq \delta\big( \min(h(n), h_{\max}(n) + 1) \big) = \delta(h(n)) \leq c\gamma^{h(n)} \leq c^{\frac{d+1}{d}} \left( \frac{C}{1 - \gamma^d} \right)^{1/d} n^{-\frac{1-\varepsilon}{d}}.$$

Now, if $d = 0$ then $n \leq C h_{\max}(n) \sum_{l=0}^{h(n)} \delta(l)^{-d} = C h_{\max}(n)(h(n) + 1)$, thus for the choice $h_{\max}(n) = \sqrt{n}$ we deduce that the loss decreases as:

$$r_n \leq \delta\big( \min(h(n), h_{\max}(n) + 1) \big) \leq c\gamma^{\sqrt{n}\min(1/C,1)-1}. \qquad \square$$

**Remark 3.** *The maximal depth function $h_{\max}(t)$ is still a parameter of the algorithm, which somehow influences the behavior of the algorithm (deep versus broad exploration of the tree). However, for a large class of problems (e.g. when $d > 0$) the choice of the order $\varepsilon$ does not impact the asymptotic performance of the algorithm.*

**Remark 4.** *Since our algorithm does not depend on $\ell$,* **our analysis is actually true for *any* semi-metric $\ell$ that satisfies Assumptions 1-4, thus Theorem 2 and Corollary 2 hold for the best possible choice of such a $\ell$.** *In particular, we can think of problems for which there exists a semi-metric $\ell$ such that the corresponding near-optimality dimension $d$ is $0$. Instead of describing a general class of problems satisfying this property, we illustrate in the next subsection non-trivial optimization problems in $\mathcal{X} = \mathbb{R}^D$ where there exists $\ell$ such that $d = 0$.*

### 4.3 Examples

**Example 1:** Consider the previous Example 1 where $\mathcal{X} = [-1,1]^D$ and $f$ is the function $f(x) = 1 - \|x\|_\infty^\alpha$, where $\alpha \geq 1$ is unknown. We have seen that DOO with the metric $\ell(x,y) = \|x - y\|_\infty^\beta$ provides a polynomial loss $r_n = O\big(n^{-\frac{1}{D}\frac{\alpha\beta}{\alpha-\beta}}\big)$ whenever $\beta < \alpha$, and an exponential loss $r_n \leq 2^{1-n}$ when $\beta = \alpha$. However, here $\alpha$ is unknown.

Now consider the SOO algorithm with the maximum depth function $h_{\max}(t) = \sqrt{t}$. As mentioned before, SOO does not require $\ell$, thus we can apply the analysis for any $\ell$ that satisfies Assumptions 1-4. So let us consider $\ell(x,y) = \|x - y\|_\infty^\alpha$. Then $\delta(h) = 2^{-h\alpha}$, $\nu = 1$, and the near-optimality dimension of $f$ under $\ell$ is $d = 0$ (and $C = 1$). We deduce that the loss of SOO is $r_n \leq 2^{(1-\sqrt{n})\alpha}$. Thus SOO provides a stretched-exponential loss without requiring the knowledge of $\alpha$.

Note that a uniform grid provides the loss $n^{-\alpha/D}$, which is polynomially decreasing only (and subject to the curse of dimensionality). Thus, in this example SOO is always better than both Uniform and DOO except if one knows perfectly $\alpha$ and would use DOO with $\beta = \alpha$ (in which case we obtain an exponential loss). The fact that SOO is not as good as DOO optimally fitted comes from the truncation of SOO at a maximal depth $h_{\max}(n) = \sqrt{n}$ (whereas DOO optimally fitted would explore the tree up to a depth linear in $n$).

**Example 2:** The same conclusion holds for Example 2, where we consider a function $f$ defined on $[0,1]^D$ that is locally equivalent to $\|x - x^*\|^\alpha$, for some unknown $\alpha > 0$ (see the precise assumptions in Section 3.3). We have seen that DOO using $\ell(x,y) = c\|x - y\|^\beta$ with $\beta < \alpha$ has a loss $r_n = O\big(n^{-\frac{1}{D}\frac{\alpha\beta}{\alpha-\beta}}\big)$, and when $\alpha = \beta$, then $d = 0$ and the loss is $r_n = O(2^{-\alpha(n/C-1)})$.

Now by using SOO (which does not require the knowledge of $\alpha$) with $h_{\max}(t) = \sqrt{t}$ we deduce the stretched-exponential loss $r_n = O(2^{-\sqrt{n}\alpha/C})$ (by using $\ell(x,y) = \|x - y\|^\alpha$ in the analysis, which gives $\delta(h) = 2^{-h\alpha}$ and $d = 0$).

### 4.4 Comparison with the DIRECT algorithm

The DIRECT (DIviding RECTangles) algorithm [JPS93, FK04, Gab01] is a Lipschitz optimization algorithm where the Lipschitz constant $L$ of $f$ is unknown. It uses an optimistic splitting technique similar to ours where at each round, it expands the set of nodes that have the highest upper-bound (as defined in DOO) for at least some value of $L$. To the best of our knowledge, there is no finite-time analysis of this algorithm (only the consistency property $\lim_{n\to\infty} r_n = 0$ is proven in [FK04]). Our approach generalizes DIRECT and we are able to derive finite-time loss bounds in a much broader setting where the function is only locally smooth and the space is semi-metric.

We are not aware of other finite-time analysis of global optimization algorithms that do not require the knowledge of the smoothness of the function.

## 5 Conclusions

We presented two algorithms: DOO requires the knowledge of the semi-metric $\ell$ under which the function $f$ is locally smooth (according to Assumption 2). SOO does not require this knowledge and performs almost as well as DOO optimally-fitted (i.e. for the best choice of $\ell$ satisfying Assumptions 1-4). We reported finite-time loss bounds using the near-optimality dimension $d$, which relates the local smoothness of $f$ around its maximum and the quantity of near-optimal states, measured by the semi-metric $\ell$. We provided illustrative examples of the performance of SOO in Euclidean spaces where the local smoothness of $f$ is unknown.

Possible future research directions include (i) deriving problem-dependent lower bounds, (ii) characterizing classes of functions $f$ such that there exists a semi-metric $\ell$ for which $f$ is locally smooth w.r.t. $\ell$ and whose corresponding near-optimal dimension is $d = 0$ (in order to have a stretched-exponentially decreasing loss), and (iii) extending the SOO algorithm to stochastic $\mathcal{X}$-armed bandits (optimization of a noisy function) when the smoothness of $f$ is unknown.

**Acknowledgements:** French ANR EXPLO-RA (ANR-08-COSI-004) and the European project COMPLACS (FP7, grant agreement n$^o$231495).

# References

[ABM10]   J.-Y. Audibert, S. Bubeck, and R. Munos. Best arm identification in multi-armed bandits. In *Conference on Learning Theory*, 2010.

[ACBF02]  P. Auer, N. Cesa-Bianchi, and P. Fischer. Finite-time analysis of the multiarmed bandit problem. *Machine Learning Journal*, 47(2-3):235–256, 2002.

[AOS07]   P. Auer, R. Ortner, and Cs. Szepesvári. Improved rates for the stochastic continuum-armed bandit problem. *20th Conference on Learning Theory*, pages 454–468, 2007.

[BM10]    S. Bubeck and R. Munos. Open loop optimistic planning. In *Conference on Learning Theory*, 2010.

[BMS09]   S. Bubeck, R. Munos, and G. Stoltz. Pure exploration in multi-armed bandits problems. In *Proc. of the 20th International Conference on Algorithmic Learning Theory*, pages 23–37, 2009.

[BMSB11]  L. Busoniu, R. Munos, B. De Schutter, and R. Babuska. Optimistic planning for sparsely stochastic systems. In *IEEE International Symposium on Adaptive Dynamic Programming and Reinforcement Learning*, 2011.

[BMSS08]  S. Bubeck, R. Munos, G. Stoltz, and Cs. Szepesvári. Online optimization of X-armed bandits. In D. Koller, D. Schuurmans, Y. Bengio, and L. Bottou, editors, *Advances in Neural Information Processing Systems*, volume 22, pages 201–208. MIT Press, 2008.

[BMSS11]  S. Bubeck, R. Munos, G. Stoltz, and Cs. Szepesvári. X-armed bandits. *Journal of Machine Learning Research*, 12:1655–1695, 2011.

[BSY11]   S. Bubeck, G. Stoltz, and J. Y. Yu. Lipschitz bandits without the Lipschitz constant. In *Proceedings of the 22nd International Conference on Algorithmic Learning Theory*, 2011.

[CM07]    P.-A. Coquelin and R. Munos. Bandit algorithms for tree search. In *Uncertainty in Artificial Intelligence*, 2007.

[FK04]    D. E. Finkel and C. T. Kelley. Convergence analysis of the direct algorithm. Technical report, North Carolina State University, Center for, 2004.

[Flo99]   C.A. Floudas. *Deterministic Global Optimization: Theory, Algorithms and Applications*. Kluwer Academic Publishers, Dordrecht / Boston / London, 1999.

[Gab01]   J. M. X. Gablonsky. *Modifications of the direct algorithm*. PhD thesis, 2001.

[GWMT06]  S. Gelly, Y. Wang, R. Munos, and O. Teytaud. Modification of UCT with patterns in monte-carlo go. Technical report, INRIA RR-6062, 2006.

[Han92]   E.R. Hansen. *Global Optimization Using Interval Analysis*. Marcel Dekker, New York, 1992.

[HM08]    J-F. Hren and R. Munos. Optimistic planning of deterministic systems. In European Workshop on Reinforcement Learning Springer LNAI 5323, editor, *Recent Advances in Reinforcement Learning*, pages 151–164, 2008.

[HT96]    R. Horst and H. Tuy. *Global Optimization ? Deterministic Approaches*. Springer, Berlin / Heidelberg / New York, 3rd edition, 1996.

[JPS93]   D. R. Jones, C. D. Perttunen, and B. E. Stuckman. Lipschitzian optimization without the lipschitz constant. *Journal of Optimization Theory and Applications*, 79(1):157–181, 1993.

[Kea96]   R. B. Kearfott. *Rigorous Global Search: Continuous Problems*. Kluwer Academic Publishers, Dordrecht / Boston / London, 1996.

[Kle04]   R. Kleinberg. Nearly tight bounds for the continuum-armed bandit problem. In *18th Advances in Neural Information Processing Systems*, 2004.

[KS06]    L. Kocsis and Cs. Szepesvári. Bandit based Monte-Carlo planning. In *Proceedings of the 15th European Conference on Machine Learning*, pages 282–293, 2006.

[KSU08]   R. Kleinberg, A. Slivkins, and E. Upfal. Multi-armed bandits in metric spaces. In *Proceedings of the 40th ACM Symposium on Theory of Computing*, 2008.

[Neu90]   Neumaier. *Interval Methods for Systems of Equations*. Cambridge University Press, 1990.

[Pin96]   J.D. Pintér. *Global Optimization in Action (Continuous and Lipschitz Optimization: Algorithms, Implementations and Applications)*. Kluwer Academic Publishers, 1996.

[SKKS10]  Niranjan Srinivas, Andreas Krause, Sham Kakade, and Matthias Seeger. Gaussian process optimization in the bandit setting: No regret and experimental design. In *International Conference on Machine Learning*, pages 1015–1022, 2010.

[Sli11]   A. Slivkins. Multi-armed bandits on implicit metric spaces. In *Advances in Neural Information Processing Systems*, 2011.

[SS00]    R.G. Strongin and Ya.D. Sergeyev. *Global Optimization with Non-Convex Constraints: Sequential and Parallel Algorithms*. Kluwer Academic Publishers, Dordrecht / Boston / London, 2000.

